# Online Submodular Set Cover, Ranking, and Repeated Active Learning

**Andrew Guillory**
Department of Computer Science
University of Washington
guillory@cs.washington.edu

**Jeff Bilmes**
Department of Electrical Engineering
University of Washington
bilmes@ee.washington.edu

## Abstract

We propose an online prediction version of submodular set cover with connections to ranking and repeated active learning. In each round, the learning algorithm chooses a sequence of items. The algorithm then receives a monotone submodular function and suffers loss equal to the cover time of the function: the number of items needed, when items are selected in order of the chosen sequence, to achieve a coverage constraint. We develop an online learning algorithm whose loss converges to approximately that of the best sequence in hindsight. Our proposed algorithm is readily extended to a setting where multiple functions are revealed at each round and to bandit and contextual bandit settings.

## 1 Problem

In an online ranking problem, at each round we choose an ordered list of items and then incur some loss. Problems with this structure include search result ranking, ranking news articles, and ranking advertisements. In search result ranking, each round corresponds to a search query and the items correspond to search results. We consider online ranking problems in which the loss incurred at each round is the number of items in the list needed to achieve some goal. For example, in search result ranking a reasonable loss is the number of results the user needs to view before they find the complete information they need. We are specifically interested in problems where the list of items is a sequence of questions to ask or tests to perform in order to learn. In this case the ranking problem becomes a repeated active learning problem. For example, consider a medical diagnosis problem where at each round we choose a sequence of medical tests to perform on a patient with an unknown illness. The loss is the number of tests we need to perform in order to make a confident diagnosis. We propose an approach to these problems using a new online version of submodular set cover.

A set function $F(S)$ defined over a ground set $V$ is called *submodular* if it satisfies the following diminishing returns property: for every $A \subseteq B \subseteq V \setminus \{v\}$, $F(A+v) - F(A) \geq F(B+v) - F(B)$. Many natural objectives measuring information, influence, and coverage turn out to be submodular [1, 2, 3]. A set function is called *monotone* if for every $A \subseteq B$, $F(A) \leq F(B)$ and *normalized* if $F(\emptyset) = 0$. *Submodular set cover* is the problem of selecting an $S \subseteq V$ minimizing $|S|$ under the constraint that $F(S) \geq 1$ where $F$ is submodular, monotone, and normalized (note we can always rescale $F$). This problem is NP-hard, but a greedy algorithm gives a solution with cost less than $1 + \ln 1/\epsilon$ that of the optimal solution where $\epsilon$ is the smallest non-zero gain of $F$ [4].

We propose the following online prediction version of submodular set cover, which we simply call *online submodular set cover*. At each time step $t = 1 \ldots T$ we choose a sequence of elements $S^t = (v_1^t, v_2^t, \ldots v_n^t)$ where each $v_i^t$ is chosen from a ground set $V$ of size $n$ (we use a superscript for rounds of the online problem and a subscript for other indices). After choosing $S^t$, an adversary reveals a submodular, monotone, normalized function $F^t$, and we suffer loss $\ell(F^t, S^t)$ where

$$\ell(F^t, S^t) \triangleq \min(\{n\} \cup \{i : F^t(S_i^t) \geq 1\}_i) \tag{1}$$

and $S_i^t \triangleq \bigcup_{j \leq i} \{v_j^t\}$ is defined to be the set containing the first $i$ elements of $S^t$ (let $S_0^t \triangleq \emptyset$). Note $\ell$ can be equivalently written $\ell(F^t, S^t) \triangleq \sum_{i=0}^n I(F^t(S_i^t) < 1)$ where $I$ is the indicator function. Intuitively, $\ell(F^t, S^t)$ corresponds to a bounded version of cover time: it is the number of items up to $n$ needed to achieve $F^t(S) \geq 1$ when we select items in the order specified by $S^t$. Thus, if coverage is not achieved, we suffer a loss of $n$. We assume that $F^t(V) \geq 1$ (therefore coverage is achieved if $S^t$ does not contain duplicates) and that the sequence of functions $(F^t)_t$ is chosen in advance (by an oblivious adversary). The goal of our learning algorithm is to minimize the total loss $\sum_t \ell(F^t, S^t)$.

To make the problem clear, we present it first in its simplest, full information version. However, we will later consider more complex variations including (1) a version where we only produce a list of length $k \leq n$ instead of $n$, (2) a multiple objective version where a set of objectives $F_1^t, F_2^t, \ldots F_m^t$ is revealed each round, (3) a bandit (partial information) version where we do not get full access to $F^t$ and instead only observe $F^t(S_1^t), F^t(S_2^t), \ldots F^t(S_n^t)$, and (4) a contextual bandit version where there is some context associated with each round.

We argue that online submodular set cover, as we have defined it, is an interesting and useful model for ranking and repeated active learning problems. In a search result ranking problem, after presenting search results to a user we can obtain implicit feedback from this user (e.g., clicks, time spent viewing each result) to determine which results were actually relevant. We can then construct an objective $F^t(S)$ such that $F^t(S) \geq 1$ iff $S$ covers or summarizes the relevant results. Alternatively, we can avoid explicitly constructing an objective by considering the bandit version of the problem where we only observe the values $F^t(S_i^t)$. For example, if the user clicked on $k$ total results then we can let $F(S_i^t) \triangleq c_i/k$ where $c_i \leq k$ is the number of results in the subset $S_i$ which were clicked. Note that the user may click an arbitrary set of results in an arbitrary order, and the user's decision whether or not to click a result may depend on previously viewed and clicked results. All that we assume is that there is some unknown submodular function explaining the click counts. If the user clicks on a small number of very early results, then coverage is achieved quickly and the ordering is desirable. This coverage objective makes sense if we assume that the set of results the user clicked are of roughly equal importance and together summarize the results of interest to the user.

In the medical diagnosis application, we can define $F^t(S)$ to be proportional to the number of candidate diseases which are eliminated after performing the set of tests $S$ on patient $t$. If we assume that a particular test result always eliminates a fixed set of candidate diseases, then this function is submodular. Specifically, this objective is the reduction in the size of the version space [5, 6]. Other active learning problems can also be phrased in terms of satisfying a submodular coverage constraint including problems that allow for noise [7]. Note that, as in the search result ranking problem, $F^t$ is not initially known but can be inferred after we have chosen $S^t$ and suffered loss $\ell(F^t, S^t)$.

## 2 Background and Related Work

Recently, Azar and Gamzu [8] extended the $O(\ln 1/\epsilon)$ greedy approximation algorithm for submodular set cover to the more general problem of minimizing the average cover time of a set of objectives. Here $\epsilon$ is the smallest non-zero gain of all the objectives. Azar and Gamzu [8] call this problem ranking with submodular valuations. More formally, we have a known set of functions $F_1, F_2, \ldots, F_m$ each with an associated weight $w_i$. The goal is then to choose a permutation $S$ of the ground set $V$ to minimize $\sum_{i=1}^m w_i \ell(F_i, S)$. The offline approximation algorithm for ranking with submodular valuations will be a crucial tool in our analysis of online submodular set cover. In particular, this offline algorithm can viewed as constructing the best single permutation $S$ for a sequence of objectives $F^1, F^2 \ldots F^T$ in hindsight (i.e., after all the objectives are known). Recently the ranking with submodular valuations problem was extended to metric costs [9].

Online learning is a well-studied problem [10]. In one standard setting, the online learning algorithm has a collection of actions $\mathcal{A}$, and at each time step $t$ the algorithm picks an action $S^t \in \mathcal{A}$. The learning algorithm then receives a loss function $\ell^t$, and the algorithm incurs the loss value for the action it chose $\ell^t(S^t)$. We assume $\ell^t(S^t) \in [0, 1]$ but make no other assumptions about the form of loss. The performance of an online learning algorithm is often measured in terms of *regret*, the difference between the loss incurred by the algorithm and the loss of the best single fixed action in hindsight: $R = \sum_{t=1}^T \ell^t(S^t) - \min_{S \in \mathcal{A}} \sum_{t=1}^T \ell^t(S)$. There are randomized algorithms which guarantee $\mathbb{E}[R] \leq \sqrt{T \ln |\mathcal{A}|}$ for *adversarial* sequences of loss functions [11]. Note that because

$\mathbb{E}[R] = o(T)$ the per round regret approaches zero. In the *bandit* version of this problem the learning algorithm only observes $\ell^t(S^t)$ [12].

Our problem fits in this standard setting with $\mathcal{A}$ chosen to be the set of all ground set permutations $(v_1, v_2, \ldots v_n)$ and $\ell^t(S^t) \triangleq \ell(F^t, S^t)/n$. However, in this case $\mathcal{A}$ is very large so standard online learning algorithms which keep weight vectors of size $|\mathcal{A}|$ cannot be directly applied. Furthermore, our problem generalizes an NP-hard offline problem which has no polynomial time approximation scheme, so it is not likely that we will be able to derive any efficient algorithm with $o(T \ln |\mathcal{A}|)$ regret. We therefore instead consider $\alpha$-regret, the loss incurred by the algorithm as compared to $\alpha$ times the best fixed prediction. $R_\alpha = \sum_{t=1}^{T} \ell^t(S^t) - \alpha \min_{S \in \mathcal{A}} \sum_{t=1}^{T} \ell^t(S)$. $\alpha$-regret is a standard notion of regret for online versions of NP-hard problems. If we can show $R_\alpha$ grows sub linearly with $T$ then we have shown loss converges to that of an offline approximation with ratio $\alpha$.

Streeter and Golovin [13] give online algorithms for the closely related problems of submodular function maximization and min-sum submodular set cover. In online submodular function maximization, the learning algorithm selects a set $S^t$ with $|S^t| \le k$ before $F^t$ is revealed, and the goal is to maximize $\sum_t F^t(S^t)$. This problem differs from ours in that our problem is a loss minimization problem as opposed to an objective maximization problem. Online min-sum submodular set cover is similar to online submodular set cover except the loss is not cover time but rather

$$\hat{\ell}(F^t, S^t) \triangleq \sum_{i=0}^{n} \max(1 - F^t(S_i^t), 0). \tag{2}$$

Min-sum submodular set cover penalizes $1 - F^t(S_i^t)$ where submodular set cover uses $I(F^t(S_i^t) < 1)$. We claim that for certain applications the hard threshold makes more sense. For example, in repeated active learning problems minimizing $\sum_t \ell(F^t, S^t)$ naturally corresponds to minimizing the number of questions asked. Minimizing $\sum_t \hat{\ell}(F^t, S^t)$ does not have this interpretation as it charges less for questions asked when $F^t$ is closer to $1$. One might hope that minimizing $\ell$ could be reduced to or shown equivalent to minimizing $\hat{\ell}$. This is not likely to be the case, as the approximation algorithm of Streeter and Golovin [13] does not carry over to online submodular set cover. Their online algorithm is based on approximating an offline algorithm which greedily maximizes $\sum_t \min(F^t(S), 1)$. Azar and Gamzu [8] show that this offline algorithm, which they call the cumulative greedy algorithm, does not achieve a good approximation ratio for average cover time.

Radlinski et al. [14] consider a special case of online submodular function maximization applied to search result ranking. In their problem the objective function is assumed to be a binary valued submodular function with $1$ indicating the user clicked on at least one document. The goal is then to maximize the number of queries which receive at least one click. For binary valued functions $\hat{\ell}$ and $\ell$ are the same, so in this setting minimizing the number of documents a user must view before clicking on a result is a min-sum submodular set cover problem. Our results generalize this problem to minimizing the number of documents a user must view before some possibly non-binary submodular objective is met. With non-binary objectives we can incorporate richer implicit feedback such as multiple clicks and time spent viewing results. Slivkins et al. [15] generalize the results of Radlinski et al. [14] to a metric space bandit setting.

Our work differs from the online set cover problem of Alon et al. [16]; this problem is a single set cover problem in which the items that need to be covered are revealed one at a time. Kakade et al. [17] analyze general online optimization problems with linear loss. If we assume that the functions $F^t$ are all taken from a known finite set of functions $\mathcal{F}$ then we have linear loss over a $|\mathcal{F}|$ dimensional space. However, this approach gives poor dependence on $|\mathcal{F}|$.

## 3  Offline Analysis

In this work we present an algorithm for online submodular set cover which extends the offline algorithm of Azar and Gamzu [8] for the ranking with submodular valuations problem. Algorithm 1 shows this offline algorithm, called the adaptive residual updates algorithm. Here we use $T$ to denote the number of objective functions and superscript $t$ to index the set of objectives. This notation is chosen to make the connection to the proceeding online algorithm clear: our online algorithm will approximately implement Algorithm 1 in an online setting, and in this case the set of objectives in

**Algorithm 1** Offline Adaptive Residual

**Input:** Objectives $F^1, F^2, \ldots F^T$
**Output:** Sequence $S_1 \subset S_2 \subset \ldots S_n$
$\quad S_0 \leftarrow \emptyset$
$\quad$ **for** $i \leftarrow 1 \ldots n$ **do**
$\quad\quad v \leftarrow \underset{v \in V}{\operatorname{argmax}} \sum_t \delta(F^t, S_{i-1}, v)$
$\quad\quad S_i \leftarrow S_{i-1} + v$
$\quad$ **end for**

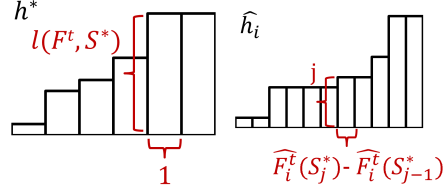

Figure 1: Histograms used in offline analysis

the offline algorithm will be the sequence of objectives in the online problem. The algorithm is a greedy algorithm similar to the standard algorithm for submodular set cover. The crucial difference is that instead of a normal gain term of $F^t(S + v) - F^t(S)$ it uses a relative gain term

$$\delta(F^t, S, v) \triangleq \begin{cases} \min(\frac{F^t(S+v) - F^t(S)}{1 - F^t(S)}, 1) & \text{if } F(S) < 1 \\ 0 & \text{otherwise} \end{cases}$$

The intuition is that (1) a small gain for $F^t$ matters more if $F^t$ is close to being covered ($F^t(S)$ close to 1) and (2) gains for $F^t$ with $F^t(S) \geq 1$ do not matter as these functions are already covered. The main result of Azar and Gamzu [8] is that Algorithm 1 is approximately optimal.

**Theorem 1** ([8]). *The loss $\sum_t \ell(F^t, S)$ of the sequence produced by Algorithm 1 is within $4(\ln(1/\epsilon) + 2)$ of that of any other sequence.*

We note Azar and Gamzu [8] allow for weights for each $F^t$. We omit weights for simplicity. Also, Azar and Gamzu [8] do not allow the sequence $S$ to contain duplicates while we do–selecting a ground set element twice has no benefit but allowing them will be convenient for the online analysis. The proof of Theorem 1 involves representing solutions to the submodular ranking problem as histograms. Each histogram is defined such that the area of the histogram is equal to the loss of the corresponding solution. The approximate optimality of Algorithm 1 is shown by proving that the histogram for the solution it finds is approximately contained within the histogram for the optimal solution. In order to convert Algorithm 1 into an online algorithm, we will need a stronger version of Theorem 1. Specifically, we will need to show that when there is some additive error in the greedy selection rule Algorithm 1 is still approximately optimal.

For the optimal solution $S^* = \operatorname{argmin}_{S \in V^n} \sum_t \ell(F^t, S)$ ($V^n$ is the set of all length $n$ sequences of ground set elements), define a histogram $h^*$ with $T$ columns, one for each function $F^t$. Let the $t$th column have with width 1 and height equal to $\ell(F^t, S^*)$. Assume that the columns are ordered by increasing cover time so that the histogram is monotone non-decreasing. Note that the area of this histogram is exactly the loss of $S^*$.

For a sequence of sets $\emptyset = S_0 \subseteq S_1 \subseteq \ldots S_n$ (e.g., those found by Algorithm 1) define a corresponding sequence of truncated objectives

$$\hat{F}_i^t(S) \triangleq \begin{cases} \min(\frac{F^t(S \cup S_{i-1}) - F^t(S_{i-1})}{1 - F^t(S_{i-1})}, 1) & \text{if } F^t(S_{i-1}) < 1 \\ 1 & \text{otherwise} \end{cases}$$

$\hat{F}_i^t(S)$ is essentially $F^t$ except with (1) $S_{i-1}$ given "for free", and (2) values rescaled to range between 0 and 1. We note that $\hat{F}_i^t$ is submodular and that if $F^t(S) \geq 1$ then $\hat{F}_i^t(S) \geq 1$. In this sense $\hat{F}_i^t$ is an easier objective than $F^t$. Also, for any $v$, $\hat{F}_i^t(\{v\}) - \hat{F}_i^t(\emptyset) = \delta(F^t, S_{i-1}, v)$. In other words, the gain of $\hat{F}_i^t$ at $\emptyset$ is the normalized gain of $F^t$ at $S_{i-1}$. This property will be crucial.

We next define truncated versions of $h^*$: $\hat{h}_1, \hat{h}_2, \ldots \hat{h}_n$ which correspond to the loss of $S^*$ for the easier covering problems involving $\hat{F}_i^t$. For each $j \in 1 \ldots n$, let $\hat{h}_i$ have $T$ columns of height $j$ with the $t$th such column of width $\hat{F}_i^t(S_j^*) - \hat{F}_i^t(S_{j-1}^*)$ (some of these columns may have 0 width). Assume again the columns are ordered by height. Figure 1 shows $h^*$ and $\hat{h}_i$.

We assume without loss of generality that $F^t(S_n^*) \geq 1$ for every $t$ (clearly some choice of $S^*$ contains no duplicates, so under our assumption that $F^t(V) \geq 1$ we also have $F^t(S_n^*) \geq 1$). Note

that the total width of $\hat{h}_i$ is then the number of functions remaining to be covered after $S_{i-1}$ is given for free (i.e., the number of $F^t$ with $F^t(S_{i-1}) < 1$). It is not hard to see that the total area of $\hat{h}_i$ is $\sum_t \hat{\ell}(\hat{F}_i^t, S^*)$ where $\hat{l}$ is the loss function for *min-sum* submodular set cover (2). From this we know $\hat{h}_i$ has area less than $h^*$. In fact, Azar and Gamzu [8] show the following.

**Lemma 1** ([8]). *$\hat{h}_i$ is completely contained within $h^*$ when $\hat{h}_i$ and $h^*$ are aligned along their lower right boundaries.*

We need one final lemma before proving the main result of this section. For a sequence $S$ define $Q_i = \sum_t \delta(F^t, S_{i-1}, v_i)$ to be the total normalized gain of the $i$th selected element and let $\Delta_i = \sum_{j=i}^n Q_j$ be the sum of the normalized gains from $i$ to $n$. Define $\Pi_i = |\{t : F^t(S_{i-1}) < 1\}|$ to be the number of functions which are still uncovered before $v_i$ is selected (i.e., the loss incurred at step $i$). [8] show the following result relating $\Delta_i$ to $\Pi_i$.

**Lemma 2** ([8]). *For any $i$, $\Delta_i \leq (\ln 1/\epsilon + 2)\Pi_i$*

We now state and prove the main result of this section, that Algorithm 1 is approximately optimal even when the $i$th greedy selection is preformed with some additive error $R_i$. This theorem shows that in order to achieve low average cover time it suffices to *approximately* implement Algorithm 1. Aside from being useful for converting Algorithm 1 into an online algorithm, this theorem may be useful for applications in which the ground set $V$ is very large. In these situations it may be possible to approximate Algorithm 1 (e.g., through sampling). Streeter and Golovin [13] prove similar results for submodular function maximization and min-sum submodular set cover. Our result is similar, but the proof is non trivial. The loss function $\ell$ is highly non linear with respect to changes in $F^t(S_i^t)$, so it is conceivable that small additive errors in the greedy selection could have a large effect. The analysis of Im and Nagarajan [9] involves a version of Algorithm 1 which is robust to a sort of *multplicative* error in each stage of the greedy selection.

**Theorem 2.** *Let $S = (v_1, v_2, \ldots v_n)$ be any sequence for which*

$$\sum_t \delta(F^t, S_{i-1}, v_i) + R_i \geq \max_{v \in V} \sum_t \delta(F^t, S_{i-1}, v)$$

*Then $\sum_t \ell(F^t, S^t) \leq 4(\ln 1/\epsilon + 2) \sum_t \ell(F^t, S^*) + n \sum_i R_i$*

*Proof.* Let $h$ be a histogram with a column for each $\Pi_i$ with $\Pi_i \neq 0$. Let $\gamma = (\ln 1/\epsilon + 2)$. Let the $i$th column have width $(Q_i + R_i)/(2\gamma)$ and height $\max(\Pi_i - \sum_j R_j, 0)/(2(Q_i + R_i))$. Note that $\Pi_i \neq 0$ iff $Q_i + R_i \neq 0$ as if there are functions not yet covered then there is some set element with non zero gain (and vice versa). The area of $h$ is

$$\sum_{i:\Pi_i \neq 0} \frac{1}{2\gamma}(Q_i + R_i) \frac{\max(\Pi_i - \sum_j R_j, 0)}{2(Q_i + R_i)} \geq \frac{1}{4\gamma} \sum_t \ell(F^t, S) - \frac{n}{4\gamma} \sum_j R_j$$

Assume $h$ and every $\hat{h}_i$ are aligned along their lower right boundaries. We show that if the $i$th column of $h$ has non-zero area then it is contained within $\hat{h}_i$. Then, it follows from Lemma 1 that $h$ is contained within $h^*$, completing the proof.

Consider the $i$th column in $h$. Assume this column has non zero area so $\Pi_i \geq \sum_j R_j$. This column is at most $(\Delta_i + \sum_{j \geq i} R_j)/(2\gamma)$ away from the right hand boundary. To show that this column is in $\hat{h}_i$ it suffices to show that after selecting the first $k = \lfloor (\Pi_i - \sum_j R_j)/(2(Q_i + R_i)) \rfloor$ items in $S^*$ we still have $\sum_t (1 - \hat{F}_i^t(S_k^*)) \geq (\Delta_i + \sum_{j \geq i} R_j)/(2\gamma)$. The most that $\sum_t \hat{F}_i^t$ can increase through the addition of one item is $Q_i + R_i$. Therefore, using the submodularity of $\hat{F}_i^t$,

$$\sum_t \hat{F}_i^t(S_k^*) - \sum_t \hat{F}_i^t(\emptyset) \leq k(Q_i + R_i) \leq \Pi_i/2 - \sum_j R_j/2$$

Therefore $\sum_t (1 - \hat{F}_i^t(S_k^*)) \geq \Pi_i/2 + \sum_j R_j/2$ since $\sum_t (1 - \hat{F}_i^t(\emptyset)) = \Pi_i$. Using Lemma 2

$$\Pi_i/2 + \sum_j R_j/2 \geq \Delta_i/(2\gamma) + \sum_j R_j/2 \geq (\Delta_i + \sum_{j \geq i} R_j)/(2\gamma)$$

$\square$

**Algorithm 2** Online Adaptive Residual

**Input:** Integer $T$

   Initialize $n$ online learning algorithms $E_1, E_2, \ldots E_n$ with $\mathcal{A} = V$

   **for** $t = 1 \to T$ **do**

      $\forall i \in 1 \ldots n$ predict $v_i^t$ with $E_i$

      $S^t \leftarrow (v_1^t, \ldots v_n^t)$

      Receive $F^t$, pay loss $l(F^t, S^t)$

      For $E_i$, $\ell^t(v) \leftarrow (1 - \delta(F^t, S_{i-1}^t, v))$

   **end for**

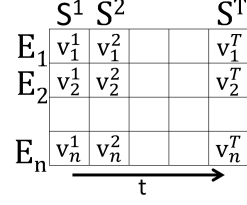

Figure 2: $E_i$ selects the $i$th element in $S^t$.

## 4   Online Analysis

We now show how to convert Algorithm 1 into an online algorithm. We use the same idea used by Streeter and Golovin [13] and Radlinski et al. [14] for online submodular function maximization: we run $n$ copies of some low regret online learning algorithm, $E_1, E_2, \ldots E_n$, each with action space $\mathcal{A} = V$. We use the $i$th copy $E_i$ to select the $i$th item in each predicted sequence $S^t$. In other words, the predictions of $E_i$ will be $v_i^1, v_i^2, \ldots v_i^T$. Figure 2 illustrates this. Our algorithm assigns loss values to each $E_i$ so that, assuming $E_i$ has low regret, $E_i$ approximately implements the $i$th greedy selection in Algorithm 1. Algorithm 2 shows this approach. Note that under our assumption that $F^1, F^2, \ldots F^T$ is chosen by an oblivious adversary, the loss values for the $i$th copy of the online algorithm are oblivious to the predictions of that run of the algorithm. Therefore we can use any algorithm for learning against an oblivious adversary.

**Theorem 3.** *Assume we use as a subroutine an online prediction algorithm with expected regret* $\mathbb{E}[R] \leq \sqrt{T \ln n}$. *Algorithm 2 has expected $\alpha$-regret* $\mathbb{E}[R_\alpha] \leq n^2 \sqrt{T \ln n}$ *for* $\alpha = 4(\ln(1/\epsilon) + 2)$

*Proof.* Define a meta-action $\tilde{v}_i$ for the sequence of actions chosen by $E_i$, $\tilde{v}_i = (v_i^1, v_i^2, \ldots v_i^T)$. We can extend the domain of $F^t$ to allow for meta-actions $F^t(S \cup \{\hat{v}_i\}) = F^t(S \cup \{v_i^t\})$. Let $\tilde{S}$ be the sequence of meta actions $\tilde{S} = (\tilde{v}_1, \tilde{v}_2, \ldots \tilde{v}_n)$. Let $R_i$ be the regret of $E_i$. Note that from the definition of regret and our choice of loss values we have that

$$\max_{v \in V} \sum_t \delta(F^t, \tilde{S}_{i-1}, v) - \sum_t \delta(F^t, \tilde{S}_{i-1}, \tilde{v}_i) = R_i$$

Therefore, $\tilde{S}$ approximates the greedy solution in the sense required by Theorem 2. Theorem 2 did not require that $S$ be constructed $V$. From Theorem 2 we then have

$$\sum_t \ell(F^t, S^t) = \sum_t \ell(F^t, \tilde{S}) \leq \alpha \sum_t \ell(F^t, S^*) + n \sum_i R_i$$

The expected $\alpha$-regret is then $\mathbb{E}[n \sum_i R_i] \leq n^2 \sqrt{T \ln n}$     $\square$

We describe several variations and extensions of this analysis, some of which mirror those for related work [13, 14, 15].

**Avoiding Duplicate Items** Since each run of the online prediction algorithm is independent, Algorithm 2 may select the same ground set element multiple times. This drawback is easy to fix. We can simply select any arbitrary $v_i \notin S_{i-1}$ if $E_i$ selects a $v_i \in S_{i-i}$. This modification does not affect the regret guarantee as selecting a $v_i \in S_{i-1}$ will always result in a gain of zero (loss of 1).

**Truncated Loss** In some applications we only care about the first $k$ items in the sequence $S^t$. For these applications it makes sense to consider a truncated version of $l(F^t, S^t)$ with parameter $k$

$$\ell^k(F^t, S^t) \triangleq \min\big(\{k\} \cup \{|S_i^t| : F^t(S_i^t) \geq 1\}\big)$$

This is cover time computed up to the $k$th element in $S^t$. The analysis for Theorem 2 also shows $\sum_t \ell^k(F^t, S^t) \leq 4(\ln 1/\epsilon + 2) \sum_t \ell(F^t, S^*) + k \sum_{i=1}^k R_i$. The corresponding regret bound is then

$k^2\sqrt{T \ln n}$. Note here we are bounding truncated loss $\sum_t \ell^k(F^t, S^t)$ in terms of untruncated loss $\sum_t \ell(F^t, S^*)$. In this sense this bound is weaker. However, we replace $n^2$ with $k^2$ which may be much smaller. Algorithm 2 achieves this bound simultaneously for all $k$.

**Multiple Objectives per Round** Consider a variation of online submodular set cover in which instead of receiving a single objective $F^t$ each round we receive a batch of objectives $F_1^t, F_2^t, \ldots F_m^t$ and incur loss $\sum_{i=1}^m \ell(F_i^t, S^t)$. In other words, each rounds corresponds to a ranking with submodular valuations problem. It is easy to extend Algorithm 2 to this setting by using $1 - (1/m)\sum_{i=1}^m \delta(F_i^t, S_{i-1}^t, v)$ for the loss of action $v$ in $E_i$. We then get $O(k^2\sqrt{mL^* \ln n} + k^2 m \ln n)$ total regret where $L^* = \sum_{t=1}^T \sum_{i=1}^m \ell(F_i^t, S^*)$ (Section 2.6 of [10]).

**Bandit Setting** Consider a setting where instead of receiving full access to $F^t$ we only observe the sequence of objective function values $F^t(S_1^t), F^t(S_2^t), \ldots F^t(S_n^t)$ (or in the case of multiple objectives per round, $F_i^t(S_j^t)$ for every $i$ and $j$). We can extend Algorithm 2 to this setting using a nonstochastic multiarmed bandits algorithm [12]. We note duplicate removal becomes more subtle in the bandit setting: should we feedback a gain of zero when a duplicate is selected or the gain of the non-duplicate replacement? We propose either is valid if replacements are chosen obliviously.

**Bandit Setting with Expert Advice** We can further generalize the bandit setting to the contextual bandit setting [18] (e.g., the bandit setting with expert advice [12]). Say that we have access at time step $t$ to predictions from a set of $m$ experts. Let $\tilde{v}_j$ be the meta action corresponding to the sequence of predictions from the $j$th expert and $\tilde{V}$ be the set of all $\tilde{v}_j$. Assume that $E_i$ guarantees low regret with respect to $\tilde{V}$

$$\sum_t \delta(F^t, S_{i-1}^t, v_i^t) + R_i \geq \max_{\tilde{v} \in \tilde{V}} \sum_t \delta(F^t, S_{i-1}^t, \tilde{v}) \tag{3}$$

where we have extended the domain of each $F^t$ to include meta actions as in the proof of Theorem 3. Additionally assume that $F^t(\tilde{V}) \geq 1$ for every $t$. In this case we can show $\sum_t \ell^k(F^t, S^t) \leq \min_{S^* \in \tilde{V}^m} \sum_t \ell^m(F^t, S^*) + k\sum_{i=1}^k R_i$. The Exp4 algorithm [12] has $R_i = O(\sqrt{nT \ln m})$ giving total regret $O(k^2\sqrt{nT \ln m})$. Experts may use context in forming recommendations. For example, in a search ranking problem the context could be the query.

## 5 Experimental Results

### 5.1 Synthetic Example

We present a synthetic example for which the online cumulative greedy algorithm [13] fails, based on the example in Azar and Gamzu [8] for the offline setting. Consider an online ad placement problem where the ground set $V$ is a set of available ad placement actions (e.g., a $v \in V$ could correspond to placing an ad on a particular web page for a particular length of time). On round $t$, we receive an ad from an advertiser, and our goal is to acquire $\lambda$ clicks for the ad using as few advertising actions as possible. Define $F^t(S_i^t)$ to be $\min(c_i^t, \lambda)/\lambda$ where $c_i^t$ is number of clicks acquired from the ad placement actions $S_i^t$.

Say that we have $n$ advertising actions of two types: 2 broad actions and $n - 2$ narrow actions. Say that the ads we receive are also of two types. Common type ads occur with probability $(n-1)/n$ and receive 1 and $\lambda - 1$ clicks respectively from the two broad actions and 0 clicks from narrow actions. Uncommon type ads occur with probability $1/n$ and receive $\lambda$ clicks from one randomly chosen narrow action and 0 clicks from all other actions. Assume $\lambda \geq n^2$. Intuitively broad actions could correspond to ad placements on sites for which many ads are relevant. The optimal strategy giving an average cover time $O(1)$ is to first select the two broad actions covering all common ads then select the narrow actions in any order. However, the offline cumulative greedy algorithm will pick all narrow actions before picking the broad action with gain 1 giving average cover time $O(n)$.

The left of Figure 3 shows average cover time for our proposed algorithm and the cumulative greedy algorithm of [13] on the same sequences of random objectives. For this example we use $n = 25$ and the bandit version of the problem with the Exp3 algorithm [12]. We also plot the average cover times for offline solutions as baselines. As seen in the figure, the cumulative algorithms converge to higher average cover times than the adaptive residual algorithms. Interestingly, the online cumulative algorithm does better than the offline cumulative algorithm: it seems added randomization helps.

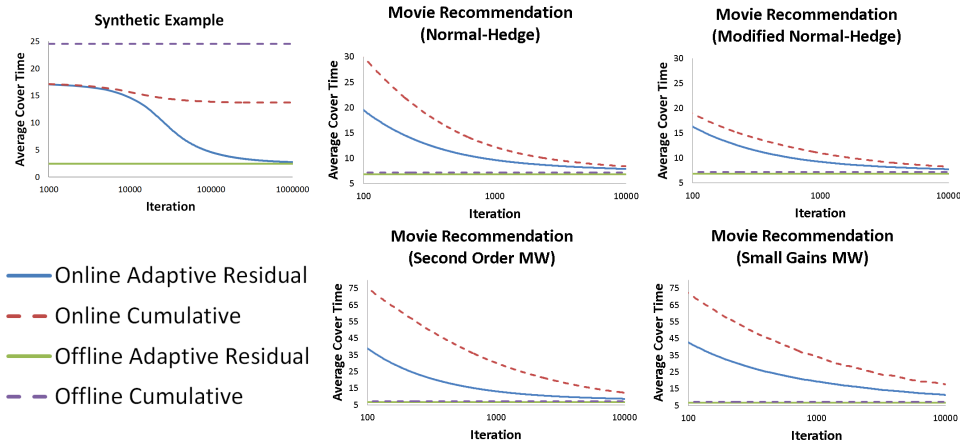

Figure 3: Average cover time

## 5.2 Repeated Active Learning for Movie Recommendation

Consider a movie recommendation website which asks users a sequence of questions before they are given recommendations. We define an online submodular set cover problem for choosing sequences of questions in order to quickly eliminate a large number of movies from consideration. This is similar conceptually to the diagnosis problem discussed in the introduction. Define the ground set $V$ to be a set of questions (for example "Do you want to watch something released in the past 10 years?" or "Do you want to watch something from the Drama genre?"). Define $F^t(S)$ to be proportional to the number of movies eliminated from consideration after asking the $t$th user $S$. Specifically, let $H$ be the set of all movies in our database and $V^t(S)$ be the subset of movies consistent with the $t$th user's responses to $S$. Define $F^t(S) \triangleq \min(|H \setminus V^t(S)|/c, 1)$ where $c$ is a constant. $F^t(S) \geq$ iff after asking the set of questions $S$ we have eliminated at least $c$ movies.

We set $H$ to be a set of 11634 movies available on Netflix's Watch Instantly service and use 803 questions based on those we used for an offline problem [7]. To simulate user responses to questions, on round $t$ we randomly select a movie from $H$ and assume the $t$th user answers questions consistently with this movie. We set $c = |H| - 500$ so the goal is to eliminate about 95% of all movies. We evaluate in the full information setting: this makes sense if we assume we receive as feedback the movie the user actually selected. As our online prediction subroutine we tried Normal-Hedge [19], a second order multiplicative weights method [20], and a version of multiplicative weights for small gains using the doubling trick (Section 2.6 of [10]). We also tried a heuristic modification of Normal-Hedge which fixes $c_t = 1$ for a fixed, more aggressive learning rate than theoretically justified. The right of Figure 3 shows average cover time for 100 runs of $T = 10000$ iterations. Note the different scale in the bottom row–these methods performed significantly worse than Normal-Hedge. The online cumulative greedy algorithm converges to a average cover time close to but slightly worse than that of the adaptive greedy method. The differences are more dramatic for prediction subroutines that converge slowly. The modified Normal-Hedge has no theoretical justification, so it may not generalize to other problems. For the modified Normal-Hedge the final average cover times are 7.72 adaptive and 8.22 cumulative. The offline values are 6.78 and 7.15.

## 6 Open Problems

It is not yet clear what practical value our proposed approach will have for web search result ranking. A drawback to our approach is that we pick a fixed order in which to ask questions. For some problems it makes more sense to consider adaptive strategies [5, 6].

### Acknowledgments

This material is based upon work supported in part by the National Science Foundation under grant IIS-0535100, by an Intel research award, a Microsoft research award, and a Google research award.

# References

[1] H. Lin and J. Bilmes. A class of submodular functions for document summarization. In *HLT*, 2011.

[2] D. Kempe, J. Kleinberg, and É. Tardos. Maximizing the spread of influence through a social network. In *KDD*, 2003.

[3] A. Krause, A. Singh, and C. Guestrin. Near-optimal sensor placements in Gaussian processes: Theory, efficient algorithms and empirical studies. *JMLR*, 2008.

[4] L.A. Wolsey. An analysis of the greedy algorithm for the submodular set covering problem. *Combinatorica*, 2(4), 1982.

[5] D. Golovin and A. Krause. Adaptive submodularity: A new approach to active learning and stochastic optimization. In *COLT*, 2010.

[6] Andrew Guillory and Jeff Bilmes. Interactive submodular set cover. In *ICML*, 2010.

[7] Andrew Guillory and Jeff Bilmes. Simultaneous learning and covering with adversarial noise. In *ICML*, 2011.

[8] Yossi Azar and Iftah Gamzu. Ranking with Submodular Valuations. In *SODA*, 2011.

[9] S. Im and V. Nagarajan. Minimum Latency Submodular Cover in Metrics. *ArXiv e-prints*, October 2011.

[10] N. Cesa-Bianchi and G. Lugosi. *Prediction, learning, and games*. Cambridge University Press, 2006.

[11] Y. Freund and R. Schapire. A desicion-theoretic generalization of on-line learning and an application to boosting. In *Computational learning theory*, pages 23–37, 1995.

[12] P. Auer, N. Cesa-Bianchi, Y. Freund, and R.E. Schapire. The nonstochastic multiarmed bandit problem. *SIAM Journal on Computing*, 32(1):48–77, 2003.

[13] M. Streeter and D. Golovin. An online algorithm for maximizing submodular functions. In *NIPS*, 2008.

[14] F. Radlinski, R. Kleinberg, and T. Joachims. Learning diverse rankings with multi-armed bandits. In *ICML*, 2008.

[15] A. Slivkins, F. Radlinski, and S. Gollapudi. Learning optimally diverse rankings over large document collections. In *ICML*, 2010.

[16] N. Alon, B. Awerbuch, and Y. Azar. The online set cover problem. In *STOC*, 2003.

[17] Sham M. Kakade, Adam Tauman Kalai, and Katrina Ligett. Playing games with approximation algorithms. In *STOC*, 2007.

[18] J. Langford and T. Zhang. The epoch-greedy algorithm for contextual multi-armed bandits. In *NIPS*, 2007.

[19] K. Chaudhuri, Y. Freund, and D. Hsu. A parameter-free hedging algorithm. In *NIPS*, 2009.

[20] N. Cesa-Bianchi, Y. Mansour, and G. Stoltz. Improved second-order bounds for prediction with expert advice. *Machine Learning*, 2007.

